# Comparing the Effects of Different Weight Distributions on Finding Sparse Representations

**David Wipf**   and   **Bhaskar Rao** *
Department of Electrical and Computer Engineering
University of California, San Diego, CA 92093
dwipf@ucsd.edu, brao@ece.ucsd.edu

## Abstract

Given a redundant dictionary of basis vectors (or atoms), our goal is to find maximally sparse representations of signals. Previously, we have argued that a sparse Bayesian learning (SBL) framework is particularly well-suited for this task, showing that it has far fewer local minima than other Bayesian-inspired strategies. In this paper, we provide further evidence for this claim by proving a restricted equivalence condition, based on the distribution of the nonzero generating model weights, whereby the SBL solution will equal the maximally sparse representation. We also prove that if these nonzero weights are drawn from an approximate Jeffreys prior, then with probability approaching one, our equivalence condition is satisfied. Finally, we motivate the worst-case scenario for SBL and demonstrate that it is still better than the most widely used sparse representation algorithms. These include Basis Pursuit (BP), which is based on a convex relaxation of the $\ell_0$ (quasi)-norm, and Orthogonal Matching Pursuit (OMP), a simple greedy strategy that iteratively selects basis vectors most aligned with the current residual.

## 1   Introduction

In recent years, there has been considerable interest in finding sparse signal representations from redundant dictionaries [1, 2, 3, 4, 5]. The canonical form of this problem is given by,

$$\min_{\boldsymbol{w}} \|\boldsymbol{w}\|_0, \qquad \text{s.t. } \boldsymbol{t} = \Phi\boldsymbol{w}, \tag{1}$$

where $\Phi \in \mathbb{R}^{N \times M}$ is a matrix whose columns represent an overcomplete or redundant basis (i.e., rank$(\Phi) = N$ and $M > N$), $\boldsymbol{w} \in \mathbb{R}^M$ is the vector of weights to be learned, and $\boldsymbol{t}$ is the signal vector. The cost function being minimized represents the $\ell_0$ (quasi)-norm of $\boldsymbol{w}$ (i.e., a count of the nonzero elements in $\boldsymbol{w}$).

Unfortunately, an exhaustive search for the optimal representation requires the solution of up to $\binom{M}{N}$ linear systems of size $N \times N$, a prohibitively expensive procedure for even modest values of $M$ and $N$. Consequently, in practical situations there is a need for approximate procedures that efficiently solve (1) with high probability. To date, the two most widely used choices are Basis Pursuit (BP) [1] and Orthogonal Matching Pursuit (OMP) [5]. BP is based on a convex relaxation of the $\ell_0$ norm, i.e., replacing $\|\boldsymbol{w}\|_0$ with $\|\boldsymbol{w}\|_1$, which leads to an attractive, unimodal optimization problem that can be readily solved via linear programming. In contrast, OMP is a greedy strategy that iteratively selects the basis

*This work was supported by DiMI grant 22-8376, Nissan, and NSF grant DGE-0333451.

vector most aligned with the current signal residual. At each step, a new approximant is formed by projecting $t$ onto the range of all the selected dictionary atoms.

Previously [9], we have demonstrated an alternative algorithm for solving (1) using a sparse Bayesian learning (SBL) framework [6] that maintains several significant advantages over other, Bayesian-inspired strategies for finding sparse solutions [7, 8]. The most basic formulation begins with an assumed likelihood model of the signal $t$ given weights $w$,

$$p(\boldsymbol{t}|\boldsymbol{w}) = (2\pi\sigma^2)^{-N/2} \exp\left(-\frac{1}{2\sigma^2}\|\boldsymbol{t} - \Phi\boldsymbol{w}\|_2^2\right). \tag{2}$$

To provide a regularizing mechanism, SBL uses the parameterized weight prior

$$p(\boldsymbol{w};\boldsymbol{\gamma}) = \prod_{i=1}^{M} (2\pi\gamma_i)^{-1/2} \exp\left(-\frac{w_i^2}{2\gamma_i}\right), \tag{3}$$

where $\boldsymbol{\gamma} = [\gamma_1, \ldots, \gamma_M]^T$ is a vector of $M$ hyperparameters controlling the prior variance of each weight. These hyperparameters can be estimated from the data by marginalizing over the weights and then performing ML optimization. The cost function for this task is

$$\mathcal{L}(\boldsymbol{\gamma}) = -\log \int p(\boldsymbol{t}|\boldsymbol{w})p(\boldsymbol{w};\boldsymbol{\gamma})d\boldsymbol{w} \propto \log|\Sigma_t| + \boldsymbol{t}^T \Sigma_t^{-1} \boldsymbol{t}, \tag{4}$$

where $\Sigma_t \triangleq \sigma^2 I + \Phi\Gamma\Phi^T$ and we have introduced the notation $\Gamma \triangleq \text{diag}(\boldsymbol{\gamma})$. This procedure, which can be implemented via the EM algorithm (or some other technique), is referred to as evidence maximization or type-II maximum likelihood [6]. Once $\boldsymbol{\gamma}$ has been estimated, a closed-form expression for the posterior weight distribution is available.

Although SBL was initially developed in a regression context, it can be easily adapted to handle (1) in the limit as $\sigma^2 \to 0$. To accomplish this we must reexpress the SBL iterations to handle the low noise limit. Applying various matrix identities to the EM algorithm-based update rules for each iteration, we arrive at the modified update [9]

$$\begin{aligned}
\boldsymbol{\gamma}_{(\text{new})} &= \text{diag}\left(\hat{\boldsymbol{w}}_{(\text{old})}\hat{\boldsymbol{w}}_{(\text{old})}^T + \left[I - \Gamma_{(\text{old})}^{1/2}\left(\Phi\Gamma_{(\text{old})}^{1/2}\right)^\dagger \Phi\right]\Gamma_{(\text{old})}\right) \\
\hat{\boldsymbol{w}}_{(\text{new})} &= \Gamma_{(\text{new})}^{1/2}\left(\Phi\Gamma_{(\text{new})}^{1/2}\right)^\dagger \boldsymbol{t},
\end{aligned} \tag{5}$$

where $(\cdot)^\dagger$ denotes the Moore-Penrose pseudo-inverse. Given that $\boldsymbol{t} \in \text{range}(\Phi)$ and assuming $\boldsymbol{\gamma}$ is initialized with all nonzero elements, then feasibility is enforced at every iteration, i.e., $\boldsymbol{t} = \Phi\hat{\boldsymbol{w}}$. We will henceforth refer to $\boldsymbol{w}^{\text{SBL}}$ as the solution of this algorithm when initialized at $\Gamma = I_M$ and $\hat{\boldsymbol{w}} = \Phi^\dagger \boldsymbol{t}$.[1] In [9] (which extends work in [10]), we have argued why $\boldsymbol{w}^{\text{SBL}}$ should be considered a viable candidate for solving (1).

In comparing BP, OMP, and SBL, we would ultimately like to know in what situations a particular algorithm is likely to find the maximally sparse solution. A variety of results stipulate rigorous conditions whereby BP and OMP are guaranteed to solve (1) [1, 4, 5]. All of these conditions depend explicitly on the number of nonzero elements contained in the optimal solution. Essentially, if this number is less than some $\Phi$-dependent constant $\kappa$, the BP/OMP solution is proven to be equivalent to the minimum $\ell_0$-norm solution. Unfortunately however, $\kappa$ turns out to be restrictively small and, for a fixed redundancy ratio $M/N$, grows very slowly as $N$ becomes large [3]. But in practice, both approaches still perform well even when these equivalence conditions have been grossly violated. To address this issue, a much looser bound has recently been produced for BP, dependent only on $M/N$. This bound holds for "most" dictionaries in the limit as $N$ becomes large [3], where "most"

is with respect to dictionaries composed of columns drawn uniformly from the surface of an $N$-dimensional unit hypersphere. For example, with $M/N = 2$, it is argued that BP is capable of resolving sparse solutions with roughly $0.3N$ nonzero elements with probability approaching one as $N \to \infty$.

Turning to SBL, we have neither a convenient convex cost function (as with BP) nor a simple, transparent update rule (as with OMP); however, we can nonetheless come up with an alternative type of equivalence result that is neither unequivocally stronger nor weaker than those existing results for BP and OMP. This condition is dependent on the relative magnitudes of the nonzero elements embedded in optimal solutions to (1). Additionally, we can leverage these ideas to motivate which sparse solutions are the most difficult to find. Later, we provide empirical evidence that SBL, even in this worst-case scenario, can still outperform both BP and OMP.

## 2 Equivalence Conditions for SBL

In this section, we establish conditions whereby $\boldsymbol{w}^{\text{SBL}}$ will minimize (1). To state these results, we require some notation. First, we formally define a dictionary $\Phi = [\boldsymbol{\phi}_1, \ldots, \boldsymbol{\phi}_M]$ as a set of $M$ unit $\ell_2$-norm vectors (atoms) in $\mathbb{R}^N$, with $M > N$ and $\text{rank}(\Phi) = N$. We say that a dictionary satisfies the unique representation property (URP) if every subset of $N$ atoms forms a basis in $\mathbb{R}^N$. We define $w_{(i)}$ as the $i$-th largest weight magnitude and $\bar{\boldsymbol{w}}$ as the $\|\boldsymbol{w}\|_0$-dimensional vector containing all the nonzero weight magnitudes of $\boldsymbol{w}$. The set of optimal solutions to (1) is $\mathcal{W}^*$ with cardinality $|\mathcal{W}^*|$. The *diversity* (or anti-sparsity) of each $\boldsymbol{w}^* \in \mathcal{W}^*$ is defined as $D^* \triangleq \|\boldsymbol{w}^*\|_0$.

**Result 1.** For a fixed dictionary $\Phi$ that satisfies the URP, there exists a set of $M-1$ scaling constants $\nu_i \in (0, 1]$ (i.e., strictly greater than zero) such that, for any $\boldsymbol{t} = \Phi\boldsymbol{w}'$ generated with

$$w'_{(i+1)} \leq \nu_i w'_{(i)} \qquad i = 1, \ldots, M-1, \qquad (6)$$

SBL will produce a solution that satisfies $\|\boldsymbol{w}^{\text{SBL}}\|_0 = \min(N, \|\boldsymbol{w}'\|_0)$ and $\boldsymbol{w}^{\text{SBL}} \in \mathcal{W}^*$.

Do to space limitations, the proof has been deferred to [11]. The basic idea is that, as the magnitude differences between weights increase, at any given scale, the covariance $\Sigma_t$ embedded in the SBL cost function is dominated by a single dictionary atom such that problematic local minimum are removed. The unique, global minimum in turn achieves the stated result.[2] The most interesting case occurs when $\|\boldsymbol{w}'\|_0 < N$, leading to the following:

**Corollary 1.** Given the additional restriction $\|\boldsymbol{w}'\|_0 < N$, then $\boldsymbol{w}^{\text{SBL}} = \boldsymbol{w}' \in \mathcal{W}^*$ and $|\mathcal{W}^*| = 1$, i.e., SBL will find the unique, maximally sparse representation of the signal $\boldsymbol{t}$.

See [11] for the proof. These results are restrictive in the sense that the dictionary dependent constants $\nu_i$ significantly confine the class of signals $\boldsymbol{t}$ that we may represent. Moreover, we have not provided any convenient means of computing what the different scaling constants might be. But we have nonetheless solidified the notion that SBL is most capable of recovering weights of different scales (and it must still find all $D^*$ nonzero weights no matter how small some of them may be). Additionally, we have specified conditions whereby we will find the unique $\boldsymbol{w}^*$ even when the diversity is as large as $D^* = N - 1$. The tighter BP/OMP bound from [1, 4, 5] scales as $O\left(N^{-1/2}\right)$, although this latter bound is much more general in that it is independent of the magnitudes of the nonzero weights.

In contrast, neither BP or OMP satisfy a comparable result; in both cases, simple 3D counter examples suffice to illustrate this point.[3] We begin with OMP. Assume the fol-

lowing:

$$\boldsymbol{w}^* = \begin{bmatrix} 1 \\ \epsilon \\ 0 \\ 0 \end{bmatrix} \quad \Phi = \begin{bmatrix} 0 & \frac{1}{\sqrt{2}} & 0 & \frac{1}{\sqrt{1.01}} \\ 0 & 0 & 1 & \frac{0.1}{\sqrt{1.01}} \\ 1 & \frac{1}{\sqrt{2}} & 0 & 0 \end{bmatrix} \quad \boldsymbol{t} = \Phi\boldsymbol{w}^* = \begin{bmatrix} \frac{\epsilon}{\sqrt{2}} \\ 0 \\ 1 + \frac{\epsilon}{\sqrt{2}} \end{bmatrix}, \quad (7)$$

where $\Phi$ satisfies the URP and has columns $\phi_i$ of unit $\ell_2$ norm. Given any $\epsilon \in (0,1)$, we will now show that OMP will necessarily fail to find $\boldsymbol{w}^*$. Provided $\epsilon < 1$, at the first iteration OMP will select $\phi_1$, which solves $\max_i |\boldsymbol{t}^T \phi_i|$, leaving the residual vector

$$\boldsymbol{r}_1 = \left(I - \phi_1\phi_1^T\right)\boldsymbol{t} = [\ \epsilon/\sqrt{2} \quad 0 \quad 0\ ]^T. \quad (8)$$

Next, $\phi_4$ will be chosen since it has the largest value in the top position, thus solving $\max_i |\boldsymbol{r}_1^T \phi_i|$. The residual is then updated to become

$$\boldsymbol{r}_2 = \left(I - [\ \phi_1 \quad \phi_4\ ][\ \phi_1 \quad \phi_4\ ]^T\right)\boldsymbol{t} = \frac{\epsilon}{101\sqrt{2}}[\ 1 \quad -10 \quad 0\ ]^T. \quad (9)$$

From the remaining two columns, $\boldsymbol{r}_2$ is most highly correlated with $\phi_3$. Once $\phi_3$ is selected, we obtain zero residual error, yet we did not find $\boldsymbol{w}^*$, which involves only $\phi_1$ and $\phi_2$. So for all $\epsilon \in (0,1)$, the algorithm fails. As such, there can be no fixed constant $\nu > 0$ such that if $w_{(2)}^* \equiv \epsilon \leq \nu w_{(1)}^* \equiv \nu$, we are guaranteed to obtain $\boldsymbol{w}^*$ (unlike with SBL).

We now give an analogous example for BP, where we present a feasible solution with smaller $\ell_1$ norm than the maximally sparse solution. Given

$$\boldsymbol{w}^* = \begin{bmatrix} 1 \\ \epsilon \\ 0 \\ 0 \end{bmatrix} \quad \Phi = \begin{bmatrix} 0 & 1 & \frac{0.1}{\sqrt{1.02}} & \frac{0.1}{\sqrt{1.02}} \\ 0 & 0 & \frac{-0.1}{\sqrt{1.02}} & \frac{0.1}{\sqrt{1.02}} \\ 1 & 0 & \frac{1}{\sqrt{1.02}} & \frac{1}{\sqrt{1.02}} \end{bmatrix} \quad \boldsymbol{t} = \Phi\boldsymbol{w}^* = \begin{bmatrix} \epsilon \\ 0 \\ 1 \end{bmatrix}, \quad (10)$$

it is clear that $\|\boldsymbol{w}^*\|_1 = 1 + \epsilon$. However, for all $\epsilon \in (0, 0.1)$, if we form a feasible solution using only $\phi_1$, $\phi_3$, and $\phi_4$, we obtain the alternate solution $\boldsymbol{w} = [\ (1 - 10\epsilon) \quad 0 \quad 5\sqrt{1.02}\epsilon \quad 5\sqrt{1.02}\epsilon\ ]^T$ with $\|\boldsymbol{w}\|_1 \approx 1 + 0.1\epsilon$. Since this has a smaller $\ell_1$ norm for all $\epsilon$ in the specified range, BP will necessarily fail and so again, we cannot reproduce the result for a similar reason as before.

At this point, it remains unclear what probability distributions are likely to produce weights that satisfy the conditions of Result 1. It turns out that the Jeffreys prior, given by $p(x) \propto 1/x$, is appropriate for this task. This distribution has the unique property that the probability mass assigned to any given scaling is equal. More explicitly, for any $s \geq 1$,

$$P\left(x \in \left[s^i, s^{i+1}\right]\right) \propto \log(s) \quad \forall i \in \mathbb{Z}. \quad (11)$$

For example, the probability that $x$ is between 1 and 10 equals the probability that it lies between 10 and 100 or between 0.01 and 0.1. Because this is an improper density, we define an approximate Jeffreys prior with range parameter $a \in (0, 1]$. Specifically, we say that $x \sim J(a)$ if

$$p(x) = \frac{-1}{2\log(a)x} \quad \text{for } x \in [a, 1/a]. \quad (12)$$

With this definition in mind, we present the following result.

**Result 2.** For a fixed $\Phi$ that satisfies the URP, let $\boldsymbol{t}$ be generated by $\boldsymbol{t} = \Phi\boldsymbol{w}'$, where $\boldsymbol{w}'$ has magnitudes drawn iid from $J(a)$. Then as $a$ approaches zero, the probability that we obtain a $\boldsymbol{w}'$ such that the conditions of Result 1 are satisfied approaches unity.

Again, for space considerations, we refer the reader to [11]. However, on a conceptual level this result can be understood by considering the distribution of order statistics. For

example, given $M$ samples from a uniform distribution between zero and some $\theta$, with probability approaching one, the distance between the $k$-th and $(k+1)$-th order statistic can be made arbitrarily large as $\theta$ moves towards infinity. Likewise, with the $J(a)$ distribution, the relative scaling between order statistics can be increased without bound as $a$ decreases towards zero, leading to the stated result.

**Corollary 2.** Assume that $D' < N$ randomly selected elements of $\boldsymbol{w}'$ are set to zero. Then as $a$ approaches zero, the probability that we satisfy the conditions of Corollary 1 approaches unity.

In conclusion, we have shown that a simple, (approximate) noninformative Jeffreys prior leads to sparse inverse problems that are optimally solved via SBL with high probability. Interestingly, it is this same Jeffreys prior that forms the implicit weight prior of SBL (see [6], Section 5.1). However, it is worth mentioning that other Jeffreys prior-based techniques, e.g., direct minimization of $p(\boldsymbol{w}) = \prod_i \frac{1}{|w_i|}$ subject to $\boldsymbol{t} = \Phi \boldsymbol{w}$, do *not* provide any SBL-like guarantees. Although several algorithms do exist that can perform such a minimization task (e.g., [7, 8]), they perform poorly with respect to (1) because of convergence to local minimum as shown in [9, 10]. This is especially true if the weights are highly scaled, and no nontrivial equivalence results are known to exist for these procedures.

## 3 Worst-Case Scenario

If the best-case scenario occurs when the nonzero weights are all of very different scales, it seems reasonable that the most difficult sparse inverse problem may involve weights of the same or even identical scale, e.g., $\bar{w}_1^* = \bar{w}_2^* = \ldots \bar{w}_{D^*}^*$. This notion can be formalized somewhat by considering the $\bar{\boldsymbol{w}}^*$ distribution that is furthest from the Jeffreys prior. First, we note that both the SBL cost function and update rules are independent of the overall scaling of the generating weights, meaning $\alpha \bar{\boldsymbol{w}}^*$ is functionally equivalent to $\bar{\boldsymbol{w}}^*$ provided $\alpha$ is nonzero. This invariance must be taken into account in our analysis. Therefore, we assume the weights are rescaled such that $\sum_i \bar{w}_i^* = 1$. Given this restriction, we will find the distribution of weight magnitudes that is most different from the Jeffreys prior.

Using the standard procedure for changing the parameterization of a probability density, the joint density of the constrained variables can be computed simply as

$$p(\bar{w}_1^*, \ldots, \bar{w}_{D^*}^*) \propto \frac{1}{\prod_{i=1}^{D^*} \bar{w}_i^*} \qquad \text{for} \quad \sum_{i=1}^{D^*} \bar{w}_i^* = 1, \;\; \bar{w}_i^* \geq 0, \forall i. \qquad (13)$$

From this expression, it is easily shown that $\bar{w}_1^* = \bar{w}_2^* = \ldots = \bar{w}_{D^*}^*$ achieves the global minimum. Consequently, equal weights are the absolute *least* likely to occur from the Jeffreys prior. Hence, we may argue that the distribution that assigns $\bar{w}_i^* = 1/D^*$ with probability one is furthest from the constrained Jeffreys prior.

Nevertheless, because of the complexity of the SBL framework, it is difficult to prove axiomatically that $\bar{\boldsymbol{w}}^* \sim \mathbf{1}$ is overall the most problematic distribution with respect to sparse recovery. We can however provide additional motivation for why we should expect it to be unwieldy. As proven in [9], the global minimum of the SBL cost function is guaranteed to produce some $\boldsymbol{w}^* \in \mathcal{W}^*$. This minimum is achieved with the hyperparameters $\gamma_i^* = (w_i^*)^2, \forall i$. We can think of this solution as forming a collapsed, or degenerate covariance $\Sigma_t^* = \Phi \Gamma^* \Phi^T$ that occupies a proper $D^*$-dimensional subspace of $N$-dimensional signal space. Moreover, this subspace must necessarily contain the signal vector $\boldsymbol{t}$. Essentially, $\Sigma_t^*$ proscribes infinite density to $\boldsymbol{t}$, leading to the globally minimizing solution.

Now consider an alternative covariance $\Sigma_t^\diamond$ that, although still full rank, is nonetheless ill-conditioned (flattened), containing $\boldsymbol{t}$ within its high density region. Furthermore, assume that $\Sigma_t^\diamond$ is not well aligned with the subspace formed by $\Sigma_t^*$. The mixture of two flattened, yet misaligned covariances naturally leads to a more voluminous (less dense) form

as measured by the determinant $|\alpha \Sigma_t^* + \beta \Sigma_t^\diamond|$. Thus, as we transition from $\Sigma_t^\diamond$ to $\Sigma_t^*$, we necessarily reduce the density at $t$, thereby increasing the cost function $\mathcal{L}(\gamma)$. So if SBL converges to $\Sigma_t^\diamond$ it has fallen into a local minimum.

So the question remains, what values of $\bar{w}^*$ are likely to create the most situations where this type of local minima occurs? The issue is resolved when we again consider the $D^*$-dimensional subspace determined by $\Sigma_t^*$. The volume of the covariance *within* this subspace is given by $\left| \bar{\Phi}^* \bar{\Gamma}^* \bar{\Phi}^{*T} \right|$, where $\bar{\Phi}^*$ and $\bar{\Gamma}^*$ are the basis vectors and hyperparameters associated with $\bar{w}^*$. The larger this volume, the higher the probability that other basis vectors will be suitably positioned so as to both (i), contain $t$ within the high density portion and (ii), maintain a sufficient component that is misaligned with the optimal covariance.

The maximum volume of $\left| \bar{\Phi}^* \bar{\Gamma}^* \bar{\Phi}^{*T} \right|$ under the constraints $\sum_i \bar{w}_i^* = 1$ and $\bar{\gamma}_i^* = (\bar{w}^*)_i^2$ occurs with $\bar{\gamma}_i^* = 1/(D^*)^2$, i.e., all the $\bar{w}_i^*$ are equal. Consequently, geometric considerations support the notion that deviance from the Jeffreys prior leads to difficulty recovering $w^*$. Moreover, empirical analysis (not shown) of the relationship between volume and local minimum avoidance provide further corroboration of this hypothesis.

## 4   Empirical Comparisons

The central purpose of this section is to present empirical evidence that supports our theoretical analysis and illustrates the improved performance afforded by SBL. As previously mentioned, others have established deterministic equivalence conditions, dependent on $D^*$, whereby BP and OMP are guaranteed to find the unique $w^*$. Unfortunately, the relevant theorems are of little value in assessing practical differences between algorithms. This is because, in the cases we have tested where BP/OMP equivalence is provably known to hold (e.g., via results in [1, 4, 5]), SBL always converges to $w^*$ as well.

As such, we will focuss our attention on the insights provided by Sections 2 and 3 as well as probabilistic comparisons with [3]. Given a fixed distribution for the nonzero elements of $w^*$, we will assess which algorithm is best (at least empirically) for most dictionaries relative to a uniform measure on the unit sphere as discussed.

To this effect, a number of monte-carlo simulations were conducted, each consisting of the following: First, a random, overcomplete $N \times M$ dictionary $\Phi$ is created whose entries are each drawn uniformly from the surface of an $N$-dimensional hypersphere. Next, sparse weight vectors $w^*$ are randomly generated with $D^*$ nonzero entries. Nonzero amplitudes $\bar{w}^*$ are drawn iid from an experiment-dependent distribution. Response values are then computed as $t = \Phi w^*$. Each algorithm is presented with $t$ and $\Phi$ and attempts to estimate $w^*$. In all cases, we ran 1000 independent trials and compared the number of times each algorithm failed to recover $w^*$. Under the specified conditions for the generation of $\Phi$ and $t$, all other feasible solutions $w$ almost surely have a diversity greater than $D^*$, so our synthetically generated $w^*$ must be maximally sparse. Moreover, $\Phi$ will almost surely satisfy the URP.

With regard to particulars, there are essentially four variables with which to experiment: (i) the distribution of $\bar{w}^*$, (ii) the diversity $D^*$, (iii) $N$, and (iv) $M$. In Figure 1, we display results from an array of testing conditions. In each *row* of the figure, $\bar{w}_i^*$ is drawn iid from a fixed distribution for all $i$; the first row uses $\bar{w}_i^* = 1$, the second has $\bar{w}_i^* \sim J(a = 0.001)$, and the third uses $\bar{w}_i^* \sim N(0, 1)$, i.e., a unit Gaussian. In all cases, the signs of the nonzero weights are irrelevant due to the randomness inherent in the basis vectors.

The *columns* of Figure 1 are organized as follows: The first column is based on the values $N = 50$, $D^* = 16$, while $M$ is varied from $N$ to $5N$, testing the effects of an increasing level of dictionary redundancy, $M/N$. The second fixes $N = 50$ and $M = 100$ while $D^*$ is varied from 10 to 30, exploring the ability of each algorithm to resolve an increasing number of nonzero weights. Finally, the third column fixes $M/N = 2$ and $D^*/N \approx 0.3$

while $N$, $M$, and $D^*$ are increased proportionally. This demonstrates how performance scales with larger problem sizes.

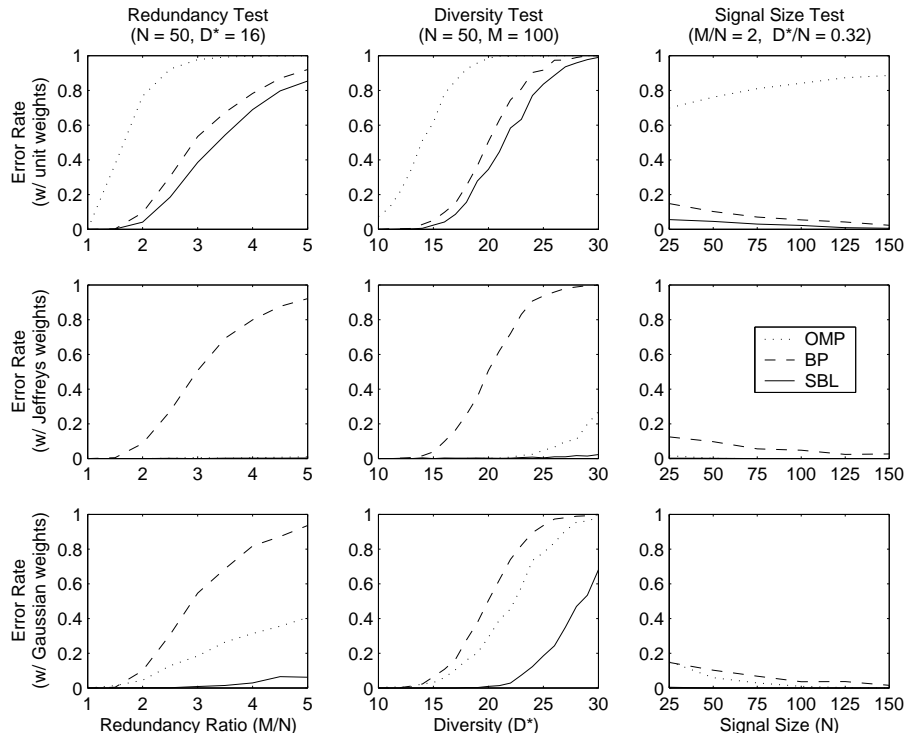

Figure 1: Empirical results comparing the probability that OMP, BP, and SBL fail to find $\boldsymbol{w}^*$ under various testing conditions. Each data point is based on 1000 independent trials. The distribution of the nonzero weight amplitudes is labeled on the far left for each row, while the values for $N$, $M$, and $D^*$ are included on the top of each column. Independent variables are labeled along the bottom of the figure.

The first row of plots essentially represents the worst-case scenario for SBL per our previous analysis, and yet performance is still consistently better than both BP and OMP. In contrast, the second row of plots approximates the best-case performance for SBL, where we see that SBL is almost infallible. The handful of failure events that do occur are because $a$ is not sufficiently small and therefore, $J(a)$ was not sufficiently close to a true Jeffreys prior to achieve perfect equivalence (see center plot). Although OMP also does well here, the parameter $a$ can generally never be adjusted such that OMP always succeeds. Finally, the last row of plots, based on Gaussian distributed weight amplitudes, reflects a balance between these two extremes. Nonetheless, SBL still holds a substantial advantage.

In general, we observe that SBL is capable of handling more redundant dictionaries (column one) and resolving a larger number of nonzero weights (column two). Also, column three illustrates that both BP and SBL are able to resolve a number of weights that grows linearly in the signal dimension ($\approx 0.3N$), consistent with the analysis in [3] (which applies only to BP). In contrast, OMP performance begins to degrade in some cases (see the upper right plot), a potential limitation of this approach. Of course additional study is necessary to fully compare the relative performance of these methods on large-scale problems.

Finally, by comparing row one, two and three, we observe that the performance of BP is roughly independent of the weight distribution, with performance slightly below the worst-

case SBL performance. Like SBL, OMP results are highly dependent on the distribution; however, as the weight distribution approaches unity, performance is unsatisfactory. In summary, while the relative proficiency between OMP and BP is contingent on experimental particulars, SBL is uniformly superior in the cases we have tested (including examples not shown, e.g., results with other dictionary types).

## 5   Conclusions

In this paper, we have related the ability to find maximally sparse solutions to the particular distribution of amplitudes that compose the nonzero elements. At first glance, it may seem reasonable that the most difficult sparse inverse problems occur when some of the nonzero weights are extremely small, making them difficult to estimate. Perhaps surprisingly then, we have shown that the exact opposite is true with SBL: The more diverse the weight magnitudes, the better the chances we have of learning the optimal solution. In contrast, unit weights offer the most challenging task for SBL. Nonetheless, even in this worst-case scenario, we have shown that SBL outperforms the current state-of-the-art; the overall assumption here being that, if worst-case performance is superior, then it is likely to perform better in a variety of situations.

For a *fixed* dictionary and diversity $D^*$, successful recovery of unit weights does not absolutely guarantee that any alternative weighting scheme will necessarily be recovered as well. However, a weaker result does appear to be feasible: For fixed values of $N$, $M$, and $D^*$, if the success rate recovering unity weights approaches one for most dictionaries, where most is defined as in Section 1, then the success rate recovering weights of any other distribution (assuming they are distributed independently of the dictionary) will also approach one. While a formal proof of this conjecture is beyond the scope of this paper, it seems to be a very reasonable result that is certainly born out by experimental evidence, geometric considerations, and the arguments presented in Section 3. Nonetheless, this remains a fruitful area for further inquiry.

## Footnotes

[1]Based on EM convergence properties, the algorithm will converge monotonically to a fixed point.

[2]Because we have effectively shown that the SBL cost function must be unimodal, etc., any proven descent method could likely be applied in place of (5) to achieve the same result.

[3]While these examples might seem slightly nuanced, the situations being illustrated can occur frequently in practice and the requisite column normalization introduces some complexity.

## References

[1]  D. Donoho and M. Elad, "Optimally sparse representation in general (nonorthogonal) dictionaries via $\ell_1$ minimization," *Proc. Nat. Acad. Sci.*, vol. 100, no. 5, pp. 2197–2202, March 2003.

[2]  R. Gribonval and M. Nielsen, "Sparse representations in unions of bases," *IEEE Transactions on Information Theory*, vol. 49, pp. 3320–3325, Dec. 2003.

[3]  D. Donoho, "For most large underdetermined systems of linear equations the minimal $\ell_1$-norm solution is also the sparsest solution," *Stanford University Technical Report*, September 2004.

[4]  J.J. Fuchs, "On sparse representations in arbitrary redundant bases," *IEEE Transactions on Information Theory*, vol. 50, no. 6, pp. 1341–1344, June 2004.

[5]  J.A. Tropp, "Greed is good: Algorithmic results for sparse approximation," *IEEE Transactions on Information Theory*, vol. 50, no. 10, pp. 2231–2242, October 2004.

[6]  M.E. Tipping, "Sparse Bayesian learning and the relevance vector machine," *Journal of Machine Learning Research*, vol. 1, pp. 211–244, 2001.

[7]  I.F. Gorodnitsky and B.D. Rao, "Sparse signal reconstruction from limited data using FOCUSS: A re-weighted minimum norm algorithm," *IEEE Transactions on Signal Processing*, vol. 45, no. 3, pp. 600–616, March 1997.

[8]  M.A.T. Figueiredo, "Adaptive sparseness using Jeffreys prior," *Advances in Neural Information Processing Systems 14*, pp. 697–704, 2002.

[9]  D.P. Wipf and B.D. Rao, "$\ell_0$-norm minimization for basis selection," *Advances in Neural Information Processing Systems 17*, pp. 1513–1520, 2005.

[10]  D.P. Wipf and B.D. Rao, "Sparse Bayesian learning for basis selection," *IEEE Transactions on Signal Processing*, vol. 52, no. 8, pp. 2153–2164, 2004.

[11]  D.P. Wipf, To appear in *Bayesian Methods for Sparse Signal Representation*, PhD Dissertation, UC San Diego, 2006 (estimated). http://dsp.ucsd.edu/~dwipf/
